# Interval Estimation for Reinforcement-Learning Algorithms in Continuous-State Domains

**Martha White**
Department of Computing Science
University of Alberta
whitem@cs.ualberta.ca

**Adam White**
Department of Computing Science
University of Alberta
awhite@cs.ualberta.ca

## Abstract

The reinforcement learning community has explored many approaches to obtaining value estimates and models to guide decision making; these approaches, however, do not usually provide a measure of confidence in the estimate. Accurate estimates of an agent's confidence are useful for many applications, such as biasing exploration and automatically adjusting parameters to reduce dependence on parameter-tuning. Computing confidence intervals on reinforcement learning value estimates, however, is challenging because data generated by the agent-environment interaction rarely satisfies traditional assumptions. Samples of value-estimates are dependent, likely non-normally distributed and often limited, particularly in early learning when confidence estimates are pivotal. In this work, we investigate how to compute robust confidences for value estimates in *continuous* Markov decision processes. We illustrate how to use *bootstrapping* to compute confidence intervals online under a changing policy (previously not possible) and prove validity under a few reasonable assumptions. We demonstrate the applicability of our confidence estimation algorithms with experiments on exploration, parameter estimation and tracking.

## 1 Introduction

In reinforcement learning, an agent interacts with the environment, learning through trial-and-error based on scalar reward signals. Many reinforcement learning algorithms estimate values for states to enable selection of maximally rewarding actions. Obtaining confidence intervals on these estimates has been shown to be useful in practice, including directing exploration [17, 19] and deciding when to exploit learned models of the environment [3]. Moreover, there are several potential applications using confidence estimates, such as teaching interactive agents (using confidence estimates as feedback), adjusting behaviour in non-stationary environments and controlling behaviour in a parallel multi-task reinforcement learning setting.

Computing confidence intervals was first studied by Kaelbling for finite-state Markov decision processes (MDPs) [11]. Since this preliminary work, many model-based algorithms have been proposed for evaluating confidences for discrete-state MDPs. The extension to continuous-state spaces with model-free learning algorithms, however, has yet to be undertaken. In this work we focus on constructing confidence intervals for online model-free reinforcement learning agents.

The agent-environment interaction in reinforcement learning does not satisfy classical assumptions typically used for computing confidence intervals, making accurate confidence estimation challenging. In the discrete case, certain simplifying assumptions make classical normal intervals more appropriate; in the continuous setting, we will need a different approach.

The main contribution of this work is a method to robustly construct confidence intervals for approximated value functions in continuous-state reinforcement learning setting. We first describe *boot-*

*strapping*, a non-parametric approach to estimating confidence intervals from data. We then prove that bootstrapping can be applied to our setting, addressing challenges due to sample dependencies, changing policies and non-stationarity (because of learning). Then, we discuss how to address complications in computing confidence intervals for sparse or local linear representations, common in reinforcement learning, such as tile coding, radial basis functions, tree-based representations and sparse distributed memories. Finally, we propose several potential applications of confidence intervals in reinforcement learning and conclude with an empirical investigation of the practicality of our confidence estimation algorithm for exploration, tuning the temporal credit parameter and tracking.

## 2 Related Work

Kaelbling was the first to employ confidence interval estimation method for exploration in finite-state MDPs [11]. The agent estimates the probability of receiving a reward of 1.0 for a given state-action pair and constructs an upper confidence bound on this estimate using a Bernoulli confidence interval. Exploration is directed by selecting the action with the highest upper confidence bound, which corresponds to actions for which it has high uncertainty or high value estimates [11].

Interval estimation for model-based reinforcement learning with discrete state spaces has been quite extensively studied. Mannor *et al.* (2004) investigated confidence estimates for the parameters of the learned transition and reward models, assuming Gaussian rewards [5, 16]. The Model Based Interval Estimation Algorithm (MBIE) uses upper confidence bounds on the model transition probabilities to select the model that gives the maximal reward [22]. The Rmax algorithm uses a heuristic notion of confidence (state visitiation counts) to determine when to explore, or exploit the learned model [3]. Both Rmax and MBIE are guaranteed to converge to the optimal policy in polynomially many steps. These guarantees, however, become difficult for continuous state spaces.

A recently proposed framework, KWIK ("Knows What It Knows"), is a formal framework for algorithms that explore efficiently by minimizing the number of times an agent must return the response "I do not know" [23]. For example, for reinforcement learning domains, KWIK-RMAX biases exploration toward states that the algorithm currently does not "know" an accurate estimate of the value [23]. KWIK-RMAX provides an uncertainty estimate (not a confidence interval) on a linear model by evaluating if the current feature vector is contained in the span of previously observed feature vectors. Though quite general, the algorithm remains theoretical due to the requirement of a solution to the model.

Bayesian methods (e.g., GPTD [6]) provide a natural measure of confidence: one can use the posterior distribution to form credible intervals for the mean value of a state-action pair. However, if one wants to use non-Gaussian priors and likelihoods, then the Bayesian approach is intractable without appropriate approximations. Although this approach is promising, we are interested in computing classical frequentist confidence intervals for agents, while not restricting the underlying learning algorithm to use a model or particular update mechanism.

Several papers have demonstrated the empirical benefits of using heuristic confidence estimates to bias exploration [14, 17, 19] and guide data collection in model learning [9, 18]. For example, Nouri *et al.* [19] discretize the state space with a KD-tree and mark the state as "known" after reaching a visitation count threshold.

In the remainder of this work, we provide the first study of estimating confidence intervals for model-free, online reinforcement learning value estimates in the continuous-state setting.

## 3 Background

In this section, we will introduce the reinforcement learning model of sequential decision making and bootstrapping, a family of techniques used to compute confidence intervals for means of dependent data from an unknown (likely non-normal) underlying distribution.

### 3.1 Reinforcement Learning

In reinforcement learning, an agent interacts with its environment, receiving observations and selecting actions to maximize a scalar reward signal provided by the environment. This interaction is

usually modeled by a Markov decision process (MDP). An MDP consists of $(S, A, P, R)$ where $S$ is the set of states; $A$ is a finite set of actions; $P$, the transition function, which describes the probability of reaching a state $s'$ from a given state and action $(s, a)$; and finally the reward function $R(s, a, s')$, which returns a scalar value for transitioning from state-action $(s, a)$ to state $s'$. The state of the environment is said to be *Markov* if $Pr(s_{t+1}, r_{t+1}|s_t, a_t) = Pr(s_{t+1}, r_{t+1}|s_t, a_t, \ldots, s_0, a_0)$. The agent's objective is to learn a *policy*, $\pi : S \to A$, such that $R$ is maximized for all $s \in S$.

Many reinforcement learning algorithms maintain an *state-action value function*, $Q^\pi(s, a)$, equal to the expected discounted sum of future rewards for a given state-action pair: $Q^\pi(s, a) = E_\pi \left[ \sum_{k=0}^\infty \gamma^k r_{t+k+1} | s_t = s, a_t = a \right]$, where $\gamma \in [0\ 1]$ discounts the contribution of future rewards. The optimal state-action value function, $Q^*(s, a)$, is the maximum achievable value given the agent starts in state $s$ and selects action $a$. The optimal policy, $\pi^*$, is greedy with respect to the optimal value function: $\pi^*(s) = \text{argmax}_{a \in A} Q^*(s, a)$ for all $s \in S$. During learning the agent must balance selecting actions to achieve high reward (according to $\hat{Q}(s, a)$) or selecting actions to gain more information about the environment. This is called the exploration-exploitation trade-off.

In many practical applications, the state space is too large to store in a table. In this case, a function approximator is used to estimate the value of a state-action pair. A linear function approximator produces a value prediction using a linear combination of basis units: $\hat{Q}(s, a) = \boldsymbol{\theta}^T \boldsymbol{\phi}(s, a)$. We refer the reader to the introductory text [25] for a more detailed discussion on reinforcement learning.

### 3.2 Bootstrapping a confidence interval for dependent data

*Bootstrapping* is a statistical procedure for estimating the distribution of a statistic (such as the sample mean), particularly when the underlying distribution is complicated or unknown, samples are dependent and power calculations (e.g. variance) are estimated with limited sample sizes [21]. This estimate can then be used to approximate a $1 - \alpha$ confidence interval around the statistic: an interval for which the probability of seeing the statistic outside of the interval is low (probability $\alpha$). For example, for potentially dependent data sampled from an unknown distribution $P(X_1, X_2, \ldots)$, we can use bootstrapping to compute a confidence interval around the mean, $T_n = n^{-1} \sum_{i=1}^n x_n$.

The key idea behind bootstrapping is that the data is an appropriate approximation, $P_n$, of the true distribution: resampling from the data represents sampling from $P_n$. Samples are "drawn" from $P_n$ to produce a bootstrap sample, $x_1^*, \ldots, x_n^* \subset \{x_1, \ldots, x_n\}$, and an estimate, $T_n^*$, of the statistic. This process is repeated $B$ times, giving $B$ samples of the statistic, $T_{n,1}^*, \ldots, T_{n,B}^*$. These, for example, can be used to estimate $\text{Var}_P(T_n) \approx \text{Var}_{P_n}(T_n) = \sum (T_{n,b}^* - \overline{T}_n^*)^2 / (B - 1)$.

Bootstrapped intervals have been shown to have a lower coverage error than normal intervals for dependent, non-normal data. A normal interval has a coverage error of $O(1/\sqrt{n})$, whereas bootstrapping has a coverage error of $O(n^{-3/2})$ [29]. The *coverage error* represents how quickly the estimated interval converges to the true interval: higher order coverage error indicates faster convergence[1]. Though the theoretical conditions for these guarantees are somewhat restrictive [29], bootstrapping has nevertheless proved very useful in practice for more general data [4, 21].

With the bootstrapped samples, a *percentile-t (studentized) interval* is constructed by

$$P(T \in (2T_n - T_{1-\alpha/2}^*, 2T_n - T_{\alpha/2}^*)) \geq 1 - \alpha$$

where $T_\beta^*$ is the $\beta$ sample quantile of $T_{n,1}^*, \ldots, T_{n,B}^*$. Usually, the $\beta$-quantile of an ordered population of size $n$ is the continuous sample quantile:

$$(1 - r)T_{n,j}^* + rT_{n,j+1}^* \quad \text{where} \quad j = \lfloor n\beta \rfloor + m, \ r = n\beta - j + m$$

where $m$ is dependent on quantile type, with $m = \frac{\beta+1}{3}$ common for non-normal distributions.

The remaining question is how to bootstrap from the sequence of samples. In the next section, we describe the block bootstrap, applicable to Markov processes, which we will show represents the structure of data for value estimates in reinforcement learning.

### 3.2.1 Moving Block Bootstrap

In the *moving block bootstrap* method, blocks of consecutive samples are drawn with replacement from a set of overlapping blocks, making the $k$-th block $\{x_{k-1+t} : t = 1 \ldots, l\}$. The bootstrap resample is the concatenation of $n/l$ blocks chosen randomly with replacement, making a time series of length $n$; $B$ of these concatenated resamples are used in the bootstrap estimate. The block bootstrap is appropriate for sequential processes because the blocks implicitly maintain a time-dependent structure. An common heuristic for the block length, $l$, is $n^{1/3}$ [8].

The moving block bootstrap was designed for stationary, dependent data; however, our scenario involves *nonstationary* data. Lahiri [12] proved a coverage error of $o(n^{-1/2})$ when applying the moving block bootstrap to nonstationary, dependent data, better than the normal coverage error. Fortunately, the conditions are not restrictive for our scenario, described further in the next section.

Note that there are other bootstrapping techniques applicable to sequential, dependent data with lower coverage error, such as the double bootstrap [13], block-block bootstrap [1] and Markov or Sieve bootstrap [28]. In particular, the Markov bootstrap has been shown to have a lower coverage error for Markov data than the block bootstrap under certain restricted conditions [10]. These techniques, however, have not been shown to be valid for nonstationary data.

## 4 Confidence intervals for continuous-state Markov decision processes

In this section, we present a theoretically sound approach to constructing confidence intervals for parametrized $Q(s, a)$ using bootstrapping for dependent data. We then discuss how to address sparse representations, such as tile coding, which make confidence estimation more complicated.

### 4.1 Bootstrapped Confidence Intervals for Global Representations

The goal is to compute a confidence estimate for $Q(s_t, a_t)$ on time step $t$. Assume that we are learning a parametrized value function $Q(s, a) = f(\boldsymbol{\theta}, s, a)$, with $\boldsymbol{\theta} \in \mathbb{R}^d$ and a smooth function $f : \mathbb{R}^d \times S \times A \to \mathbb{R}$. A common example is a linear value function $Q(s, a) = \boldsymbol{\theta}^T \boldsymbol{\phi}(s, a)$, with $\boldsymbol{\phi} : S \times A \to \mathbb{R}^d$. During learning, we have a sequence of changing weights, $\{\boldsymbol{\theta}_1, \boldsymbol{\theta}_2, \ldots, \boldsymbol{\theta}_n\}$ up to time step $n$, corresponding to the random process $\{\boldsymbol{\Theta}_1, \ldots, \boldsymbol{\Theta}_n\}$. If this process were stationary, then we could compute an interval around the mean of the process. In almost all cases, however, the process will be nonstationary with means $\{\mu_1, \ldots, \mu_n\}$. Instead, our goal is to estimate

$$\bar{f}_n(s, a) = n^{-1} \sum_{t=1}^{n} E[f(\boldsymbol{\Theta}_t, s, a)]$$

which represents the variability in the current estimation of the function $\hat{Q}$ for any given state-action pair, $(s, a) \in S \times A$. Because $Q$ is parametrized, the sequence of weights, $\{\boldsymbol{\Theta}_t\}$, represents the variability for the uncountably many state-action pairs.

Assume that the weight vector on time step $t + 1$ is drawn from the unknown distribution $P_a[(\boldsymbol{\Theta}_{t+1}, s_{t+1})|(\boldsymbol{\theta}_t, s_t), \ldots, (\boldsymbol{\theta}_{t-k}, s_{t-k})]$, giving a $k$-order Markov dependence on previous states and weight vectors. Notice that $P_a$ incorporates $P$ and $R$, using $s_t$, $\boldsymbol{\theta}_t$ (giving the policy $\pi$) and $R$ to determine the reward passed to the algorithm to then obtain $\boldsymbol{\theta}_{t+1}$. This allows the learning algorithm to select actions using confidence estimates based on the history of the $k$ most recent $\boldsymbol{\theta}$, without invalidating that the sequence of weights are drawn from $P_a$. In practice, the length of the dependence, $k$, can be estimated using *auto-correlation* [2].

Applying the Moving Block Bootstrap method to a non-stationary sequence of $\theta$'s requires several assumptions on the underlying MDP and the learning algorithm. We require two assumptions on the underlying MDP: a bounded density function and a strong mixing requirement. The assumptions on the algorithm are less strict, only requiring that the algorithm be non-divergent and produce a sequence of $\{Q_t(s, a)\}$ that 1) satisfy a smoothness condition (a dependent Cramer condition), 2) have a bounded twelfth moment and 3) satisfy an $m$-dependence relation where sufficiently separated $Q_i(s, a), Q_j(s, a)$ are independent. Based on these assumptions (stated formally in the supplement), we can prove that the moving block bootstrap produces an interval with a coverage error of $o(n^{-1/2}$ for the studentized interval on $f_n(s, a)$.

**Theorem 1** *Given that Assumption 1-7 are satisfied and there exists constants $C_1, C_2 > 0$, $0 < \alpha \leq \beta < 1/4$ such that $C_1 n^\alpha < l < C_2 n^\beta$ (i.e. $l$ increases with $n$), then the moving block bootstrap produces a one-sided confidence interval that is consistent and has a coverage error of $o(n^{-1/2})$ for the studentization of the mean of the process $\{f(\boldsymbol{\theta}_t, s, a)\}$, where $Q_t(s, a) = f(\boldsymbol{\theta}_t, s, a)$.*

The proof for the above theorem follows Lahiri's proof [12] for the coverage error of the moving block bootstrap for nonstationary data. The general approach for coverage error proofs involve approximating the unknown distribution with an Edgeworth expansion (see [7]), with the coverage error dependent on the order of the expansion, similar to the the idea of a Taylor series expansion.

Assuming $P_a$ is $k$-order Markov results in two important practical implications on the learning algorithm: 1) inability to use eligibility traces and 2) restrictions on updates to parameters (such as the learning rate). These potential issues, however, are actually not restrictive. First, the tail of eligibility traces has little effect, particularly for larger $k$; the most recent $k$ weights incorporate the most important information for the eligibility traces. Second, the learning rate, for example, cannot be updated based on time. The learning rate, however, can still be adapted based on changes between weight vectors, a more principled approach taken, by the meta-learning algorithm, IDBD [24].

The final algorithm is summarized in the pseudocode below. In practice, a window of data of length $w$ is stored due to memory restrictions; other data selection techniques are possible. Corresponding to the notation in Section 3.2, $Q_i$ represents the data samples (of $\hat{Q}(s,a)$), $(Q_{i,1}^*, \ldots, Q^*i, M)$ the dependently sampled blocks for the $i$th resample and $T_i^*$ the mean of the $i$ resample.

---

**Algorithm 1** GetUpperConfidence($f(\cdot, s, a), \{\boldsymbol{\theta}_{n-w}, \ldots \boldsymbol{\theta}_n\}, \alpha$)

---

$l$ = block length, $B$ = num bootstrap resamples
last $w$ weights and confidence level $\alpha$ (= 0.05)

1:   $Q_N \leftarrow \{f(\boldsymbol{\theta}_{n-w}, s, a), \ldots f(\boldsymbol{\theta}_n, s, a)\}$
2:   Blocks = $\{[Q_{n-w}, \ldots, Q_{n-w+l-1}], [Q_{n-w+1}, \ldots, Q_{n-w+l}], \ldots, [Q_{n-l+1}, \ldots, Q_n]\}$
3:   $M \leftarrow \lfloor w/l \rfloor$          the number of length $l$ blocks to sample with replacement and concatenate
4:   **for all** $i = 1$ to $B$ **do**
5:     $(Q_1^*, Q_2^*, \ldots, Q_{M*l}^*) \leftarrow$ concatMRandomBlocks(Blocks, M)
6:     $T_i^* = \frac{1}{M*l} \sum Q_j^*$
7:   **end for**
8:   sort($\{T_1^*, \ldots, T_B^*\}$)
9:   $j \leftarrow \lfloor \frac{B\alpha}{2} + \frac{\alpha+2}{6} \rfloor, r \leftarrow \frac{B\alpha}{2} + \frac{\alpha+2}{6} - j$
10: $T_{\alpha/2}^* \leftarrow (1-r)T_j^* + rT_{j+1}^*$
11: Return $2\text{mean}(Q_N) - T_{\alpha/2}^*$

---

## 4.2   Bootstrapped Confidence Intervals for Sparse Representations

We have shown that bootstrapping is a principled approach for computing intervals for global representations; sparse representations, however, complicate the solution. In an extreme case, for example, for linear representations, features active on time step $t$ may have never been active before. Samples $Q_1(s_t, a_t), \ldots, Q_t(s_t, a_t)$ would therefore all equal $Q_0(s_t, a_t)$, because the weights would have never been updated for those features. Consequently, the samples erroneously indicate low variance for $Q(s_t, a_t)$.

We propose that, for sparse linear representations, the samples for the weights can be treated independently and still produce a reasonable, though currently unproven, bootstrap interval. Notice that for $\theta(i)$ the $i$th feature

$$P_a[(\boldsymbol{\theta}_t, s_t)|(\boldsymbol{\theta}_{t-1}, s_{t-1}), \ldots, (\boldsymbol{\theta}_{t-k}, s_{t-k})] = \Pi_{i=1}^d P_a[(\theta_t(i), s_t)|(\boldsymbol{\theta}_{t-1}, s_{t-1}), \ldots, (\boldsymbol{\theta}_{t-k}, s_{t-k})]$$

because updates to weights $\theta(i), \theta(j)$ are independent given the previous states and weights vectors for all $i, j \in \{1, \ldots, d\}$. We could, therefore, estimate upper confidence bounds on the individual weights, $ucb_i(s, a)$, and then combine them, via $ucb(s, a) = \sum_{i=1}^d ucb_i(s, a) * \phi_i(s, a)$, to produce an upper confidence bound on $Q(s_t, a_t)$. To approximate the variance of $\theta(i)$ on time step $t$, we can use the last $w$ samples of $\theta(i)$ where $\theta(i)$ changed.

Proving coverage error results for sparse representations will require analyzing the covariance between components of $\boldsymbol{\theta}$ over time. The above approach for sparse representations does not capture this covariance; due to sparsity, however, the dependence between many of the samples for $\theta(i)$ and $\theta(j)$ will likely be weak. We could potentially extend the theoretical results by bounding the covariance between the samples and exploiting independencies. The means for individual weights could likely be estimated separately, therefore, and still enable a valid confidence interval. In future work, a potential extension is to estimate the covariances between the individual weights to improve the interval estimate.

## 5  Applications of confidence intervals for reinforcement learning

The most obvious application of interval estimation is to bias exploration to select actions with high uncertainty. Confidence-based exploration should be comparable to optimistic initialization in domains where exhaustive search is required and find better policies in domains where noisy rewards and noisy dynamics can cause the optimistic initialization to be prematurely decreased and inhibit exploration. Furthermore, confidence-based exploration reduces parameter tuning because the policy does not require knowledge of the reward range, as in softmax and optimistic initialization.

Confidence-based exploration could be beneficial in domains where the problem dynamics and reward function change over time. In an extreme case, the agent may converge to a near-optimal policy before the goal is teleported to another portion of the space. If the agent continues to act greedily with respect to its action-value estimates without re-exploring, it may act sub-optimally indefinitely. These tracking domains require that the agent "notice" that its predictions are incorrect and begin searching for a better policy. AN example of a changing reward signals arises in interactive teaching. In this scenario, the a human teaching shapes the agent by providing a drifting reward signal. Even in stationary domains, tracking the optimal policy may be more effective than converging due to the non-stationarity introduced by imperfect function approximation [26].

Another potential application of confidence estimation is to automate parameter tuning online. For example, many TD-based reinforcement learning algorithms use an eligibility parameter ($\lambda$) to address the credit assignment problem. Learning performance can be sensitive to $\gamma$. There has been little work, however, exploring the effects of different decay functions for $\lambda$; using different $\lambda$ values for each state/feature; or for meta-learning $\lambda$. Confidence estimates could be used to increase $\lambda$ when the agent is uncertain, reflecting and decrease $\lambda$ for confident value estimates [25].

Confidence estimates could also be used to guide the behaviour policy for a parallel multi-task reinforcement learning system. Due to recent theoretical developments [15], several target value functions can be learned in parallel, off-policy, based on a single stream of data from a behaviour policy. The behaviour policy should explore to provide samples that generalize well between the various target policies, speeding overall convergence. For example, if one-sided intervals are maintained for each target value functions, the behaviour policy could select an action corresponding to the maximal sum of those intervals. Exploration is then biased to highly uncertain areas where more samples are required.

Finally, confidence estimates could be used to determine when features should be evaluated in a feature construction algorithm. Many feature construction algorithms, such as cascade correlation networks, interleave proposing candidate features and evaluation. In an online reinforcement learning setting, these methods freeze the representation for a fixed window of time to accurately evaluate the candidate [20]. Instead of using a fixed window, a more principled approach is to evaluate the features after the confidence on the weights of the candidate features reached some threshold.

## 6  Experimental Results

In this section, we provide a preliminary experimental investigation into the practicality of confidence estimation in continuous-state MDPs. We evaluate a naive implementation of the block bootstrap method for (1) exploration in a noisy reward domain, (2) automatically tuning $\lambda$ in the Cartpole domain and (3) tracking a moving goal in a navigation task. In all tests we used the Sarsa($\lambda$) learning algorithm with tile coding function approximation (see Sutton and Barto [25]). All experiments were evaluated using RL-Glue [27] and averaged over 30 independent runs.

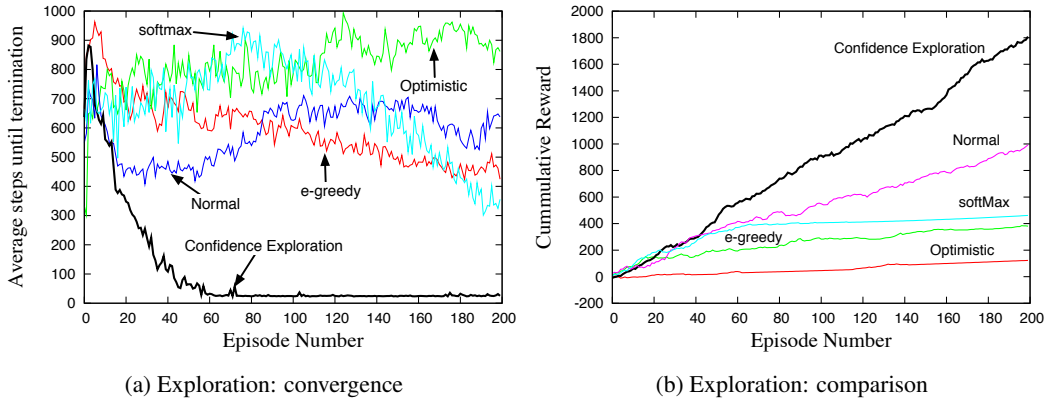

(a) Exploration: convergence          (b) Exploration: comparison

Figure 1: Results showing (a) convergence of various exploration techniques in the navigation task and (b) average cumulative reward of various exploration techniques on the navigation task.

## 6.1 Exploration

To evaluate the effectiveness of confidence-based exploration, we use a simple two-goal continuous navigation task. The *small goal* yields a reward of 1.0 on every visit. The *flashing goal* yields a reward selected uniformly from $\{100, -100, 5, -5, 50\}$. The reward on all other steps is zero and $\gamma = 0.99$ (similar results for -1 per step and $\gamma = 1.0$). The agent's observation is a continuous $(x, y)$ position and actions move the agent $\{N,S,E,W\}$ perturbed by uniform noise 10% of the time. We present only the first 200 episodes to highlight early learning performance.

Similar to Kaelbling, we select the action with the highest upper confidence in each state. We compare our confidence exploration algorithm to three baselines commonly used in continuous state MDPs: (1) $\epsilon$-greedy (selecting the highest-value action with probability $1 - \epsilon$, random otherwise), (2) optimistic initialization (initializing all weights to a high fixed value to encourage exploration) and (3) softmax (choosing actions probabilistically according to their values). We also compare our algorithm to an exploration policy using normal (instead of bootstrapped) intervals to investigate the effectiveness of making simplifying assumptions on the data distribution. We present the results for the best parameter setting for each exploration policy for clarity. Figure 1 summarizes the results.

The $\epsilon$-greedy policy convergences slowly to the small goal. The optimistic policy slowly converges to the small goal for lower initializations and does not favour either goal for higher initializations. The softmax policy navigates to the small goal on most runs and also convergences slowly. The normal-interval exploration policy does prefer the flashing goal but not as quickly as the bootstrap policy. Finally, the bootstrap-interval exploration policy achieves highest cumulative reward and is the only policy that converges to the flashing goal, despite the large variance in the reward signal.

## 6.2 Adjusting Lambda

To illustrate the effect of adjusting $\lambda$ based on confidence intervals, we study the Cartpole problem. We selected Cartpole because the performance of Sarsa is particularly sensitive to $\lambda$ in this domain. The objective in Cartpole is to apply forces to a cart on a track to keep a pole from falling over. An episode ends when the pole falls past a given angle or the cart reaches the end of the track. The reward is +1 for each step of the episode. The agent's observations are the cart position and velocity and the poles' angle and angular velocity. The Cartpole environment is based on Sutton and Barto's [25] pole-balancing task and is available in RL-library [27].

To adjust the $\lambda$ value, we reset $\lambda$ on every time step: $\lambda = \text{normalized}(ucb)$ where $ucb = 0.9 * ucb + 0.1 * \text{getUpperConfidence}(\phi(s, a), \theta, \alpha)$. The confidence estimates were only used to adjust $\lambda$ for clarity: exploration was performed using optimistic initialization. Figure 2 presents the average balancing time on the last episode for various values of $\lambda$. The flat line depicts the average balancing time for Sarsa with $\lambda$ tuned via confidence estimates. Setting $\lambda$ via confidence estimates achieves performance near the best value of $\lambda$. We also tested adjusting $\lambda$ using normal confidence intervals, however, the normal confidence intervals resulted in worse performance then any fixed value of $\lambda$.

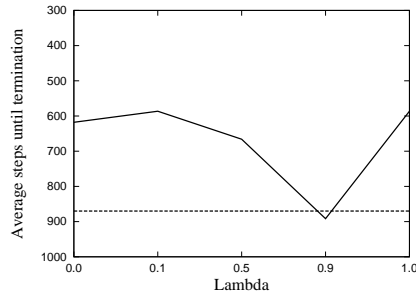

Figure 2: Performance of Sarsa($\lambda$) on Cartpole for various values of $\lambda$. The straight line depicts the performance of Sarsa with $\lambda$ adjusted using the confidence estimation algorithm.

## 6.3   Non-stationary Navigation Task

One natural source of non-stationarity is introduced by *shaping* a robot through successive approximations to a goal task (e.g., changing the reward function). We studied the effects of this form of non-stationarity, where the agent learns to go to a goal and then another, better goal becomes available (near the first goal to better guide it to the next goal). In our domain, the agent receives -1 reward per step and +10 at termination in a goal region. After 150 episodes, the goal region is teleported to a new location within 50 steps of the previous goal. The agent receives +10 in the new goal and now 0 in the old goal. We used $\epsilon = 0$ to enable exploration only with optimistic initialization.

We recorded the number of times the agent converged to the new goal with the change after an initial learning period of 150 episodes. The bootstrap-based explorer found the new goal 70% of the time. It did not always find the new goal because the -1 structure biased it to stay with the safe 0 goal. Interestingly, optimistic initialization was unable to find the new goal because of this bias, illustrating that the confidence-based explorer detected the increase in variance and promoted re-exploration automatically.

## 7   Conclusion

In this work, we investigated constructing confidence intervals on value estimates in the continuous-state reinforcement learning setting. We presented a robust approach to computing confidence estimates for function approximation using bootstrapping, a nonparametric estimation technique. We proved that our confidence estimate has low coverage error under mild assumptions on the learning algorithm. In particular, we did so even for a changing policy that uses the confidence estimates. We illustrated the usefulness of our estimates for three applications: exploration, tuning $\lambda$ and tracking.

We are currently exploring several directions for future work. We have begun testing the confidence-based exploration on a mobile robot platform. Despite the results presented in this work, many traditional deterministic, negative cost-to-goal problems (e.g., Mountain Car, Acrobot and Puddle World) are efficiently solved using optimistic exploration. Robotic tasks, however, are often more naturally formulated as continual learning tasks with a sparse reward signal, such as negative reward for bumping into objects, or a positive reward for reaching some goal. We expect confidence based techniques to perform better in these settings where the reward range may be truly unknown (e.g. generated dynamically by a human teacher) and under natural variability in the environment (noisy sensors and imperfect motion control). We have also begun evaluating confidence-interval driven behaviour for large-scale, parallel off-policy learning on the same robot platform.

There are several potential algorithmic directions, in addition to those mentioned throughout this work. We could potentially improve coverage error by extending other bootstrapping techniques, such as the Markov bootstrap, to non-stationary data. We could also explore the theoretical work on exponential bounds, such as the Azuma-Hoeffding inequality, to obtain different confidence estimates with low coverage error. Finally, it would be interesting to extend the theoretical results in the paper to sparse representations.

**Acknowledgements:** We would like to thank Csaba Szepesvári, Narasimha Prasad and Daniel Lizotte for their helpful comments and NSERC, Alberta Innovates and the University of Alberta for funding the research.

## Footnotes

[1]More theoretically, coverage error is the approximation error in the Edgeworth expansions used to approximate the distribution in bootstrap proofs.

# References

[1] D.W.K. Andrews. The block-block bootstrap: Improved asymptotic refinements. *Econometrica*, 72(3):673–700, 2004.

[2] G.E.P. Box, G.M. Jenkins, and G.C. Reinsel. *Time series analysis: forecasting and control*. Holden-day San Francisco, 1976.

[3] R. I. Brafman and M. Tennenholtz. R-max - a general polynomial time algorithm for near-optimal reinforcement learning. *Journal of Machine Learning Research*, 3:213–231, 2002.

[4] A.C. Davison and DV Hinkley. *Bootstrap methods and their application*. Cambridge Univ Pr, 1997.

[5] E. Delage and S. Mannor. Percentile optimization for Markov decision processes with parameter uncertainty. *Operations Research*, 58(1):203, 2010.

[6] Y. Engel, S. Mannor, and R. Meir. Reinforcement learning with Gaussian processes. In *Proceedings of the 22nd international conference on Machine learning*, page 208. ACM, 2005.

[7] P Hall. *The bootstrap and Edgeworth expansion*. Springer Series in Statistics, Jan 1997.

[8] Peter Hall, Joel L. Horowitz, and Bing-Yi Jing. On blocking rules for the bootstrap with dependent data. *Biometrika*, 82(3):561–74, 1995.

[9] Todd Hester and Peter Stone. Generalized model learning for reinforcement learning in factored domains. In *The Eighth International Conference on Autonomous Agents and Multiagent Systems (AAMAS)*, 2009.

[10] J.L. Horowitz. Bootstrap methods for Markov processes. *Econometrica*, 71(4):1049–1082, 2003.

[11] Leslie P. Kaelbling. *Learning in Embedded Systems (Bradford Books)*. The MIT Press, May 1993.

[12] SN Lahiri. Edgeworth correction by moving blockbootstrap for stationary and nonstationary data. *Exploring the Limits of Bootstrap*, pages 183–214, 1992.

[13] S. Lee and PY Lai. Double block bootstrap confidence intervals for dependent data. *Biometrika*, 2009.

[14] L. Li, M.L. Littman, and C.R. Mansley. Online exploration in least-squares policy iteration. In *Proc. of The 8th Int. Conf. on Autonomous Agents and Multiagent Systems*, volume 2, pages 733–739, 2009.

[15] H.R. Maei, C. Szepesvári, S. Bhatnagar, and R.S. Sutton. Toward off-policy learning control with function approximation. *ICM (2010)*, 50, 2010.

[16] S. Mannor, D. Simester, P. Sun, and J.N. Tsitsiklis. Bias and variance in value function estimation. In *Proceedings of the twenty-first international conference on Machine learning*, page 72. ACM, 2004.

[17] Lilyana Mihalkova and Raymond J. Mooney. Using active relocation to aid reinforcement learning. In *FLAIRS Conference*, pages 580–585, 2006.

[18] Peter Stone Nicholas K. Jong. Model-based exploration in continuous state spaces. In *The 7th Symposium on Abstraction, Reformulation, and Approximation*, July 2007.

[19] A. Nouri and M.L. Littman. Multi-resolution exploration in continuous spaces. In *NIPS*, pages 1209–1216, 2008.

[20] François Rivest and Doina Precup. Combining td-learning with cascade-correlation networks. In *ICML*, pages 632–639, 2003.

[21] J. Shao and D. Tu. *The jackknife and bootstrap*. Springer, 1995.

[22] A.L. Strehl and M.L. Littman. An empirical evaluation of interval estimation for markov decision processes. In *Proc. of the 16th Int. Conf. on Tools with Artificial Intelligence (ICTAI04)*, 2004.

[23] Alexander L. Strehl and Michael L. Littman. Online linear regression and its application to model-based reinforcement learning. In *NIPS*, 2007.

[24] R.S. Sutton. Adapting bias by gradient descent: An incremental version of delta-bar-delta. In *Proceedings of the National Conference on Artificial Intelligence*, pages 171–171, 1992.

[25] R.S. Sutton and A.G. Barto. *Introduction to reinforcement learning*. MIT Press Cambridge, USA, 1998.

[26] R.S. Sutton, A. Koop, and D. Silver. On the role of tracking in stationary environments. In *Proceedings of the 24th international conference on Machine learning*, page 878. ACM, 2007.

[27] Brian Tanner and Adam White. RL-Glue : Language-independent software for reinforcement-learning experiments. *JMLR*, 10:2133–2136, September 2009.

[28] B A. Turlach. Bandwidth selection in kernel density estimation: A review. In *CORE and Institut de Statistique*, 1993.

[29] J. Zvingelis. On bootstrap coverage probability with dependent data. *Computer-Aided Econ.*, 2001.

